# Estimating Car Insurance Premia: a Case Study in High-Dimensional Data Inference

**Nicolas Chapados, Yoshua Bengio, Pascal Vincent, Joumana Ghosn, Charles Dugas, Ichiro Takeuchi, Linyan Meng**

University of Montreal, dept. IRO, CP 6128, Succ. Centre-Ville, Montréal, Qc, Canada, H3C3J7

{chapados,bengioy,vincentp,ghosn,dugas,takeuchi,mengl}@iro.umontreal.ca

## Abstract

Estimating insurance premia from data is a difficult regression problem for several reasons: the large number of variables, many of which are discrete, and the very peculiar shape of the noise distribution, asymmetric with fat tails, with a large majority zeros and a few unreliable and very large values. We compare several machine learning methods for estimating insurance premia, and test them on a large data base of car insurance policies. We find that function approximation methods that do not optimize a squared loss, like Support Vector Machines regression, do not work well in this context. Compared methods include decision trees and generalized linear models. The best results are obtained with a mixture of experts, which better identifies the least and most risky contracts, and allows to reduce the median premium by charging more to the most risky customers.

## 1  Introduction

The main mathematical problem faced by actuaries is that of estimating how much each insurance contract is expected to cost. This conditional expected claim amount is called the *pure premium* and it is the basis of the gross premium charged to the insured. This expected value is conditionned on information available about the insured and about the contract, which we call *input profile* here. This regression problem is difficult for several reasons: large number of examples, large number variables (most of which are discrete and multi-valued), non-stationarity of the distribution, and a conditional distribution of the dependent variable which is very different from those usually encountered in typical applications of machine learning and function approximation. This distribution has a mass at zero: the vast majority of the insurance contracts do not yield any claim. This distribution is also strongly asymmetric and it has fat tails (on one side only, corresponding to the large claims).

In this paper we study and compare several learning algorithms along with methods traditionally used by actuaries for setting insurance premia. The study is performed on a large database of *automobile insurance* policies. The methods that were tried

are the following: the constant (unconditional) predictor as a benchmark, linear regression, generalized linear models (McCullagh and Nelder, 1989), decision tree models (CHAID (Kass, 1980)), support vector machine regression (Vapnik, 1998), multi-layer neural networks, mixtures of neural network experts, and the current premium structure of the insurance company.

In a variety of practical applications, we often find data distributions with an asymmetric heavy tail extending out towards more positive values. Modeling data with such an **asymmetric heavy-tail distribution** is essentially difficult because *outliers*, which are sampled from the tail of the distribution, have a strong influence on parameter estimation. When the distribution is *symmetric* (around the mean), the problems caused by outliers can be reduced using *robust* estimation techniques (Huber, 1982; F.R.Hampel et al., 1986; Rousseeuw and Leroy, 1987) which basically intend to ignore or downweight outliers. Note that these techniques do not work for an asymmetric distribution: most outliers are on the same side of the mean, so downweighting them introduces a strong bias on its estimation: the conditional expectation would be systematically underestimated.

There is another statistical difficulty, due to the large number of variables (mostly discrete) and the fact that many interactions exist between them. Thus the traditional actuarial methods based on tabulating average claim amounts for combinations of values are quickly hurt by the **curse of dimensionality**, unless they make hurtful independence assumptions (Bailey and Simon, 1960). Finally, there is a computational difficulty: we had access to a large database of $\approx 8 \times 10^6$ examples, and the training effort and numerical stability of some algorithms can be burdensome for such a large number of training examples.

This paper is organized as follows: we start by describing the mathematical criteria underlying insurance premia estimation (section 2), followed by a brief review of the learning algorithms that we consider in this study, including our best-performing mixture of positive-output neural networks (section 3). We then highlight our most important experimental results (section 4), and in view of them conclude with an examination of the prospects for applying statistical learning algorithms to insurance modeling (section 5).

## 2   Mathematical Objectives

The first goal of insurance premia modeling is to estimate the *expected claim amount* for a given insurance contract for a future one-year period (here we consider that the amount is 0 when no claim is filed). Let $X \in \mathbf{R}^m$ denote the customer and contract *input profile*, a vector representing all the information known about the customer and the proposed insurance policy before the beginning of the contract. Let $A \in \mathbf{R}^+$ denote the amount that the customer claims during the contract period; we shall assume that $A$ is non-negative. Our objective is to estimate this claim amount, which is the *pure premium* $p_{\mathrm{pure}}$ of a given contract $x$:[1]

$$p_{\mathrm{pure}}(x) = E[A|X = x]. \tag{1}$$

**The Precision Criterion.** In practice, of course, we have no direct access to the quantity (1), which we must estimate. One possible criterion is to seek the *most precise* estimator, which minimizes the mean-squared error (MSE) over a data set $D = \{\langle x_\ell, a_\ell \rangle\}_{\ell=1}^L$. Let $\mathcal{P} = \{p(\cdot; \theta)\}$ be a function class parametrized by the

parameter vector $\theta$. The MSE criterion produces the most precise function (on average) within the class, as measured with respect to $D$:

$$\theta^* = \arg\min_{\theta} \frac{1}{L} \sum_{i=1}^{L} (p(x_i; \theta) - a_i)^2. \tag{2}$$

Is it an appropriate criterion and why? First one should note that if $p_1$ and $p_2$ are two estimators of $E[A|X]$, then the MSE criterion is a good indication of how close they are to $E[A|X]$, since by the law of iterated expectations,

$$\begin{aligned} E[(p_1(X) - A)^2] - E[(p_2(X) - A)^2] &= E[(p_1(X) - E[A|X])^2] \\ &\quad -E[(p_2(X) - E[A|X])^2], \end{aligned}$$

and of course the expected MSE is minimized when $p(X) = E[A|X]$.

**The Fairness Criterion.** However, in insurance policy pricing, the precision criterion is not the sole part of the picture; just as important is that the estimated premia do not systematically discriminate against specific segments of the population. We call this objective the *fairness criterion*. We define the *bias of the premia* $b(P)$ to be the difference between the average premium and the average incurred amount, in a given population $P$:

$$b(P) = \frac{1}{|P|} \sum_{\langle x_i, a_i \rangle \in P} p(x_i) - a_i, \tag{3}$$

where $|P|$ denotes the cardinality of the set $P$, and $p(\cdot)$ is some premia estimation function. A possible fairness criterion would be based on minimizing the norm of the bias over every subpopulation $Q$ of $P$. From a practical standpoint, such a minimization would be extremely difficult to carry out. Furthermore, the bias over small subpopulations is hard to estimate with statistical significance. We settle instead for an approximation that gives good empirical results. After training a model to minimize the MSE criterion (2), we define a finite number of disjoint subsets (subpopulations) of the test set $P$, $P_k \subset P, P_k \cap P_{j \neq k} = \emptyset$, and *verify* that the absolute bias is not significantly different from zero. The subsets $P_k$ can be chosen at convenience; in our experiments, we considered 10 subsets of equal size delimited by the deciles of the test set premium distribution. In this way, we verify that, for example, for the group of contracts with a premium between the 5th and the 6th decile, the average premium matches the average claim amount.

## 3 Models Evaluated

An important requirement for any model of insurance premia is that it should produce *positive* premia: the company does not want to charge negative money to its customers! To obtain **positive outputs neural networks** we have considered using an exponential activation function at the output layer but this created numerical difficulties (when the argument of the exponential is large, the gradient is huge). Instead, we have successfully used the "softplus" activation function (Dugas et al., 2001):

$$\text{softplus}(s) = \log(1 + e^s)$$

where $s$ is the weighted sum of an output neuron, and softplus$(s)$ is the corresponding predicted premium. Note that this function is convex, monotone increasing, and can be considered as a smooth version of the "positive part" function $\max(0, x)$.

The best model that we obtained is a **mixture of experts** in which the experts are **positive outputs neural networks**. The *gater network* (Jacobs et al., 1991) has softmax outputs to obtain positive weights summing to one.

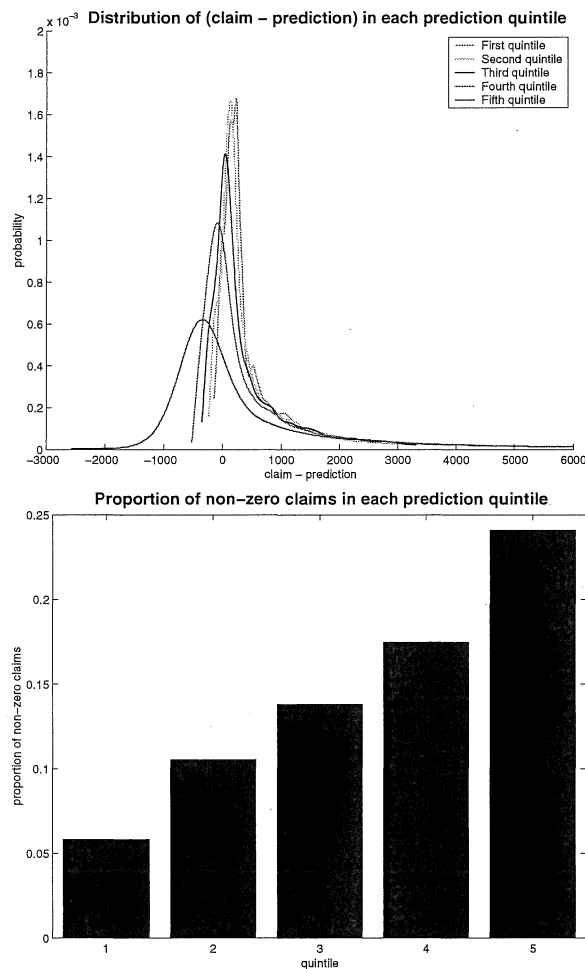

**Figure 1:** *A view of the conditional distribution of the claim amounts in the out-of-sample test set. Top: probability density of (claim amount – conditional expectation) for 5 quintiles of the conditional expectation, excluding zero-claim records. The mode moves left for increasing conditional expectation quintiles. Bottom: proportion of non-zero claim records per quintile of the prediction.*

The mixture model was compared to other models. The **constant model** only has intercepts as free parameters. The **linear model** corresponds to a ridge linear regression (with weight decay chosen with the validation set). **Generalized linear models** (GLM) estimate the conditional expectation from $f(x) = e^{b+w'x}$ with parameters $b$ and $w$. Again weight decay is used and tuned on the validation set. There are many variants of GLMs and they are popular for building insurance models, since they provide positive outputs, interpretable parameters, and can be associated to parametric models of the noise.

Decision trees are also used by practitioners in the insurance industry, in particular the **CHAID**-type models (Kass, 1980; Biggs, Ville and Suen, 1991), which use statistical criteria for deciding how to split nodes and when to stop growing the tree. We have compared our models with a CHAID implementation based on (Biggs, Ville and Suen, 1991), adapted for regression purposes using a MANOVA analysis. The threshold parameters were selected based on validation set MSE.

Regression **Support Vector Machines** (SVM) (Vapnik, 1998) were also evaluated

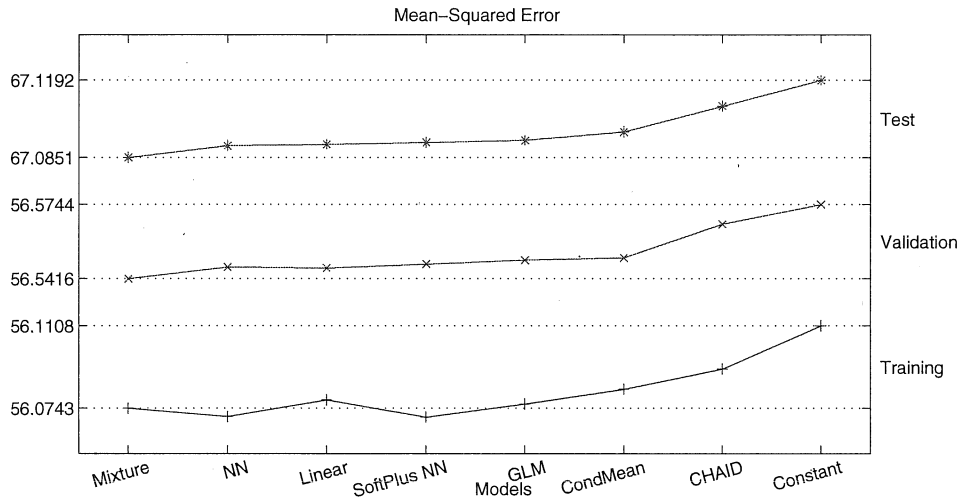

**Figure 2:** *MSE results for eight models. Models have been sorted in ascending order of test results. The training, validation and test curves have been shifted closer together for visualization purposes (the significant differences in MSE between the 3 sets are due to "outliers"). The out-of-sample test performance of the Mixture model is significantly better than any of the other. Validation based model selection is confirmed on test results. CondMean is a constructive greedy version of GLM.*

but yielded disastrous results for two reasons: (1) SVM regression optimizes an $L_1$-like criterion that finds a solution close to the conditional median, whereas the MSE criterion is minimized for the conditional mean, and because the distribution is highly asymmetric the conditional median is far from the conditional mean; (2) because the output variable is difficult to predict, the required number of support vectors is huge, also yielding poor generalization. Since the median is actually 0 for our data, we tried to train the SVM using only the cases with positive claim amounts, and compared the performance to that obtained with the GLM and the neural network. The SVM is still way off the mark because of the above two reasons. Figure 1 (top) illustrates the fat tails and asymmetry of the conditional distribution of the claim amounts.

Finally, we compared the best statistical model with a proprietary table-based and rule-based premium estimation method that was provided to us as the **benchmark** against which to judge improvements.

## 4  Experimental Results

Data from five kinds of losses were included in the study (i.e. a sub-premium was estimated for each type of loss), but we report mostly aggregated results showing the error on the total estimated premium. The input variables contain information about the policy (e.g., the date to deal with inflation, deductibles and options), the car, and the driver (e.g., about past claims, past infractions, etc...). Most variables are subject to discretization and binning. Whenever possible, the bins are chosen such that they contain approximately the same number of observations. For most models except CHAID, the discrete variables are one-hot encoded. The number of input random variables is 39, all discrete except one, but using one-hot encoding this results in an input vector $x$ of length $m = 266$. An overall data set containing about

**Table 1:** *Statistical comparison of the prediction accuracy difference between several individual learning models and the best Mixture model. The p-value is given under the null hypothesis of no difference between Model #1 and the best Mixture model. Note that* **all differences are statistically significant.**

| Model #1 | Model #2 | Mean MSE Diff. | Std. Error | $Z$ | $p$-value |
|---|---|---|---|---|---|
| Constant | Mixture | 3.40709e-02 | 3.32724e-03 | 10.2400 | **0** |
| CHAID | Mixture | 2.35891e-02 | 2.57762e-03 | 9.1515 | **0** |
| GLM | Mixture | 7.54013e-03 | 1.15020e-03 | 6.5555 | **2.77e-11** |
| Softplus NN | Mixture | 6.71066e-03 | 1.09351e-03 | 6.1368 | **4.21e-10** |
| Linear | Mixture | 5.82350e-03 | 1.32211e-03 | 4.4047 | **5.30e-06** |
| NN | Mixture | 5.23885e-03 | 1.41112e-03 | 3.7125 | **1.02e-04** |

**Table 2:** *MSE difference between benchmark and Mixture models across the 5 claim categories (kinds of losses) and the total claim amount. In all cases except category 1, the Mixture model is* **statistically significantly** *($p < 0.05$) more precise than the benchmark model.*

| Claim Category | MSE Difference | 95% Confidence Interval | |
|---|---|---|---|
| (Kind of Loss) | Benchmark minus Mixture | Lower | Higher |
| Category 1 | 20669.53 | ( −4682.83 – | 46021.89 ) |
| Category 2 | 1305.57 | ( 1032.76 – | 1578.37 ) |
| Category 3 | 244.34 | ( 6.12 – | 482.55 ) |
| Category 4 | 1057.51 | ( 623.42 – | 1491.60 ) |
| Category 5 | 1324.31 | ( 1077.95 – | 1570.67 ) |
| **Total claim amount** | 60187.60 | ( 7743.96 – | 112631.24 ) |

8 million examples is randomly permuted and split into a training set, validation set and test set, respectively of size 50%, 25% and 25% of the total. The validation set is used to select among models (including the choice of capacity), and the test set is used for final statistical comparisons. Sample-wise paired statistical tests are used to reduce the effect of huge per-sample variability.

Figure 1 is an attempt at capturing the shape of the conditional distribution of claim amounts given input profiles, by considering the distributions of claim amounts in different quantiles of the prediction (pure premium), on the test set. The top figure excludes the point mass of zero claims and rather shows the *difference* between the claim amount and the estimated conditional expectation (obtained with the mixture model). The bottom histogram shows that the fraction of claims increases nicely for the higher predicted pure premia.

Table 1 and Figure 2 summarize the comparison between the test MSE of the different tested models. *NN* is a neural network with linear output activation whereas *Softplus NN* has the *softplus* output activations. The *Mixture* is the mixture of softplus neural networks. This result identifies the mixture model with softplus neural networks as the best-performing of the tested statistical models. Our conjecture is that the mixture model works better because it is more robust to the effect of "outliers" (large claims). Classical robust regression methods (Rousseeuw and Leroy, 1987) work by discarding or downweighting outliers: they cannot be applied here because the claims distribution is highly asymmetric (the extreme values are always large ones, the claims being all non-negative). Note that the capacity of each model has been tuned on the validation set. Hence, e.g. CHAID could have easily yielded lower training error, but at the price of worse generalization.

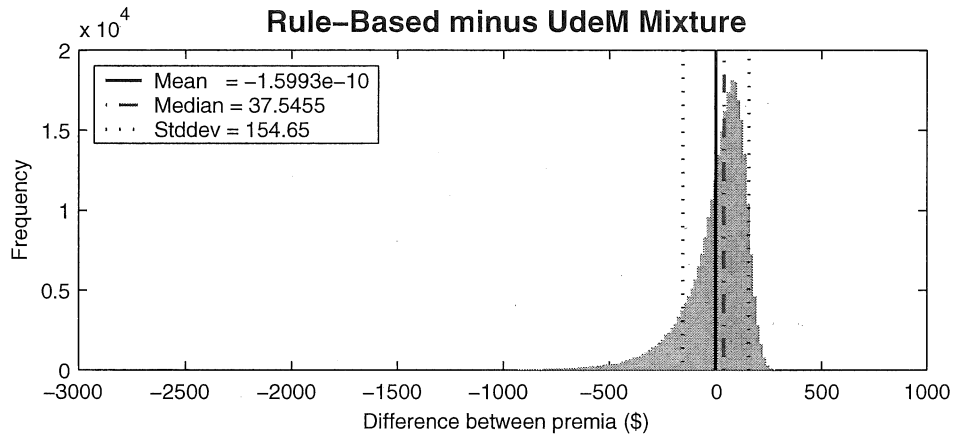

**Figure 3:** *The premia difference distribution is **negatively skewed**, but has a **positive median** for a mean of zero. This implies that the benchmark model (current pricing) undercharges risky customers, while overcharging typical customers.*

Table 2 shows a comparison of this model against the rule-based benchmark. The improvements are shown across the five types of losses. In all cases the mixture improves, and the improvement is significant in four out of the five as well as across the sum of the five.

A qualitative analysis of the resulting predicted premia shows that the mixture model has smoother and more spread-out premia than the benchmark. The analysis (figure 3) also reveals that the difference between the mixture premia and the benchmark premia is negatively skewed, with a positive median, i.e., the typical customer will pay less under the new mixture model, but the "bad" (risky) customers will pay much more.

To evaluate **fairness**, as discussed in the previous section, the distribution of premia computed by the best model is analyzed, splitting the contracts in 10 groups according to their premium level. Figure 4 shows that the premia charged are fair for each sub-population.

## 5   Conclusion

This paper illustrates a successful data-mining application in the insurance industry. It shows that a specialized model (the mixture model), that was designed taking into consideration the specific problem posed by the data (outliers, asymmetric distribution, positive outputs), performs significantly better than existing and popular learning algorithms. It also shows that such models can significantly improve over the current practice, allowing to compute premia that are lower for less risky contracts and higher for more risky contracts, thereby reducing the cost of the median contract.

Future work should investigate in more detail the role of temporal non-stationarity, how to optimize fairness (rather than just test for it afterwards), and how to further increase the robustness of the model with respect to large claim amounts.

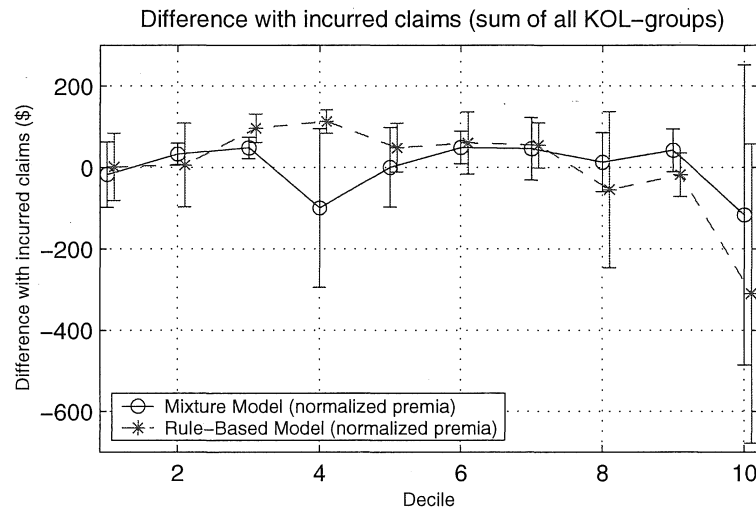

**Figure 4:** *We ensure fairness by comparing the average incurred amount and premia within each decile of the premia distribution; both models are* **generally fair** *to subpopulations. The error bars denote 95% confidence intervals. The comparison is for the sum of claim amounts over all 5 kinds of losses (KOL).*

## Footnotes

[1] The pure premium is distinguished from the premium actually charged to the customer, which must account for the risk remaining with the insurer, the administrative overhead, desired profit, and other business costs.

# References

Bailey, R. A. and Simon, L. (1960). Two studies in automobile insurance ratemaking. *ASTIN Bulletin*, 1(4):192–217.

Biggs, D., Ville, B., and Suen, E. (1991). A method of choosing multiway partitions for classification and decision trees. *Journal of Applied Statistics*, 18(1):49–62.

Dugas, C., Bengio, Y., Bélisle, F., and Nadeau, C. (2001). Incorporating second order functional knowledge into learning algorithms. In Leen, T., Dietterich, T., and Tresp, V., editors, *Advances in Neural Information Processing Systems*, volume 13, pages 472–478.

F.R.Hampel, E.M.Ronchetti, P.J.Rousseeuw, and W.A.Stahel (1986). *Robust Statistics, The Approach based on Influence Functions*. John Wiley & Sons.

Huber, P. (1982). *Robust Statistics*. John Wiley & Sons Inc.

Jacobs, R. A., Jordan, M. I., Nowlan, S. J., and Hinton, G. E. (1991). Adaptive mixture of local experts. *Neural Computation*, 3:79–87.

Kass, G. (1980). An exploratory technique for investigating large quantities of categorical data. *Applied Statistics*, 29(2):119–127.

McCullagh, P. and Nelder, J. (1989). *Generalized Linear Models*. Chapman and Hall, London.

Rousseeuw, P. and Leroy, A. (1987). *Robust Regression and Outlier Detection*. John Wiley & Sons Inc.

Vapnik, V. (1998). *Statistical Learning Theory*. Wiley, Lecture Notes in Economics and Mathematical Systems, volume 454.
